# A New Approach to Hybrid HMM/ANN Speech Recognition Using Mutual Information Neural Networks

**G. Rigoll, C. Neukirchen**

Gerhard-Mercator-University Duisburg
Faculty of Electrical Engineering
Department of Computer Science
Bismarckstr. 90, Duisburg, Germany

## ABSTRACT

This paper presents a new approach to speech recognition with hybrid HMM/ANN technology. While the standard approach to hybrid HMM/ANN systems is based on the use of neural networks as posterior probability estimators, the new approach is based on the use of mutual information neural networks trained with a special learning algorithm in order to maximize the mutual information between the input classes of the network and its resulting sequence of firing output neurons during training. It is shown in this paper that such a neural network is an optimal neural vector quantizer for a discrete hidden Markov model system trained on Maximum Likelihood principles. One of the main advantages of this approach is the fact, that such neural networks can be easily combined with HMM's of any complexity with context-dependent capabilities. It is shown that the resulting hybrid system achieves very high recognition rates, which are now already on the same level as the best conventional HMM systems with continuous parameters, and the capabilities of the mutual information neural networks are not yet entirely exploited.

## 1 INTRODUCTION

Hybrid HMM/ANN systems deal with the optimal combination of artificial neural networks (ANN) and hidden Markov models (HMM). Especially in the area of automatic speech recognition, it has been shown that hybrid approaches can lead to very powerful and efficient systems, combining the discriminative capabilities of neural networks and the superior dynamic time warping abilities of HMM's. The most popular hybrid approach is described in (Hochberg, 1995) and replaces the component modeling the emission probabilities of the HMM by a neural net. This is possible, because it is shown

in (Bourlard, 1994) that neural networks can be trained so that the output of the m-th neuron approximates the posterior probability $p(\Omega_m|\underline{x})$. In this paper, an alternative method for constructing a hybrid system is presented. It is based on the use of discrete HMM's which are combined with a neural vector quantizer (VQ) in order to form a hybrid system. Each speech feature vector is presented to the neural network, which generates a firing neuron in its output layer. This neuron is processed as VQ label by the HMM's. There are the following arguments for this alternative hybrid approach:

- The neural vector quantizer has to be trained on a special information theory criterion, based on the mutual information between network input and resulting neuron firing sequence. It will be shown that such a network is the optimal acoustic processor for a discrete HMM system, resulting in a profound mathematical theory for this approach.
- Resulting from this theory, a formula can be derived which jointly describes the behavior of the HMM and the neural acoustic processor. In that way, both systems can be described in a unified manner and both major components of the hybrid system can be trained using a unified learning criterion.
- The above mentioned theoretical background leads to the development of new neural network paradigms using novel training algorithms that have not been used before in other areas of neurocomputing, and therefore represent major challenges and issues in learning and training for neural systems.
- The neural networks can be easily combined with any HMM system of arbitrary complexity. This leads to the combination of optimally trained neural networks with very powerful HMM's, having all features useful for speech recognition, e.g. triphones, function words, crossword triphones, etc.. Context-dependency, which is very desirable but relatively difficult to realize with a pure neural approach, can be left to the HMM's.
- The resulting hybrid system has still the basic structure of a discrete system, and therefore has all the effective features associated with discrete systems, e.g. quick and easy training as well as recognition procedures, real-time capabilities, etc..
- The work presented in this paper has been also successfully implemented for a demanding speech recognition problem, the 1000 word speaker-independent continuous Resource Management speech recognition task. For this task, the hybrid system produces one of the best recognition results obtained by any speech recognition system.

In the following section, the theoretical foundations of the hybrid approach are briefly explained. A unified probabilistic model for the combined HMM/ANN system is derived, describing the interaction of the neural and the HMM component. Furthermore, it is shown that the optimal neural acoustic processor can be obtained from a special information theoretic network training algorithm.

## 2 INFORMATION THEORY PRINCIPLES FOR NEURAL NETWORK TRAINING

We are considering now a neural network of arbitrary topology used as neural vector quantizer for a discrete HMM system. If K patterns are presented to the hybrid system during training, the feature vectors resulting from these patterns using any feature extraction method can be denoted as $\underline{x}(k)$, k=1...K. If these feature vectors are presented to the input layer of a neural network, the network will generate one firing neuron for each presentation. Hence, all K presentations will generate a stream of firing neurons with length K resulting from the output layer of the neural net. This label stream is denoted as Y=y(1)...y(K). The label stream Y will be presented to the HMM's, which calculate the probability that this stream has been observed while a pattern of a certain class has been presented to the system. It is assumed, that M different classes $\Omega_m$ are active in the

system, e.g. the words or phonemes in speech recognition. Each feature vector $\underline{x}(k)$ will belong to one of these classes. The class $\Omega_m$, to which feature vector $\underline{x}(k)$ belongs is denoted as $\Omega(k)$. The major training issue for the neural network can be now formulated as follows: How should the weights of the network be trained, so that the network produces a stream of firing neurons that can be used by the discrete HMM's in an optimal way? It is known that HMM's are usually trained with information theory methods which mostly rely on the Maximum Likelihood (ML) principle. If the parameters of the hybrid system (i.e. transition and emission probabilities and network weights) are summarized in the vector $\underline{\theta}$, the probability $p_{\underline{\theta}}(\underline{x}(k)|\Omega(k))$ denotes the probability of the pattern $\underline{x}$ at discrete time k, under the assumption that it has been generated by the model representing class $\Omega(k)$, with parameter set $\underline{\theta}$. The ML principle will then try to maximize the joint probability of all presented training patterns $\underline{x}(k)$, according to the following Maximum Likelihood function:

$$\underline{\theta}^* = \arg\max_{\underline{\theta}} \left\{ \frac{1}{K} \sum_{k=1}^{K} \log p_{\underline{\theta}}(\underline{x}(k)|\Omega(k)) \right\} \tag{1}$$

where $\underline{\theta}^*$ is the optimal parameter vector maximizing this equation. Our goal is to feed the feature vector $\underline{x}$ into a neural network and to present the neural network output to the Markov model. Therefore, one has to introduce the neural network output in a suitable manner into the above formula. If the vector $\underline{x}$ is presented to the network input layer, and we assume that there is a chance that any neuron $y_n$, n=1...N (with network output layer size N) can fire with a certain probability, then the output probability $p(\underline{x}|\Omega)$ in (1) can be written as:

$$p(\underline{x}|\Omega) = \sum_{n=1}^{N} p(\underline{x}, y_n|\Omega) = \sum_{n=1}^{N} p(y_n|\Omega) \cdot p(\underline{x}|y_n, \Omega) \tag{2}$$

Now, the combination of the neural component with the HMM can be made more obvious: In (2), typically the probability $p(y_n|\Omega)$ will be described by the Markov model, in terms of the emission probabilities of the HMM. For instance, in continuous parameter HMM's, these probabilities are interpreted as weights for Gaussian mixtures. In the case of semi-continuous systems or discrete HMM's, these probabilities will serve as discrete emission probabilities of the codebook labels. The probability $p(\underline{x}|y_n, \Omega)$ describes the acoustic processor of the system and is characterizing the relation between the vector $\underline{x}$ as input to the acoustic processor and the label $y_n$, which can be considered as the n-th output component of the acoustic processor. This n-th output component may characterize e.g. the n-th Gaussian mixture component in continuous parameter HMM's, or the generation of the n-th label of a vector quantizer in a discrete system. This probability is often considered as independent of the class $\Omega$ and can then be expressed as $p(\underline{x}|y_n)$. It is exactly this probability, that can be modeled efficiently by our neural network. In this case, the vector $\underline{x}$ serves as input to the neural network and $y_n$ characterizes the n-th neuron in the output layer of the network. Using Bayes law, this probability can be written as:

$$p(\underline{x}|y_n) = \frac{p(y_n|\underline{x}) \cdot p(\underline{x})}{p(y_n)} \tag{3}$$

yielding for (2):

$$p(\underline{x}|\Omega) = p(\underline{x}) \cdot \sum_{n=1}^{N} p(y_n|\Omega) \cdot \frac{p(y_n|\underline{x})}{p(y_n)} \tag{4}$$

Using again Bayes law to express

$$p(y_n | \Omega) = \frac{p(\Omega | y_n) \cdot p(y_n)}{p(\Omega)} \tag{5}$$

one obtains from (4):

$$p(\underline{x} | \Omega) = \frac{p(\underline{x})}{p(\Omega)} \cdot \sum_{n=1}^{N} p(\Omega | y_n) \cdot p(y_n | \underline{x}) \tag{6}$$

We have now modified the class-dependent probability of the feature vector $\underline{x}$ in a way that allows the incorporation of the probability $p(y_n | \underline{x})$. This probability allows a better characterization of the behavior of the neural network, because it describes the probability of the various neurons $y_n$, if the vector $\underline{x}$ is presented to the network input. Therefore, these probabilities give a good description of the input/output behavior of the neural network. Eq. (6) can therefore be considered as probabilistic model for the hybrid system, where the neural acoustic processor is characterized by its input/output behavior. Two cases can be now distinguished: In the first case, the neural network is assumed to be a probabilistic paradigm, where each neuron fires with a certain probability, if an input vector is presented. In this case all neurons contribute to the information forwarded to the HMM's. As already mentioned, in this paper, the second possible case is considered, namely that only one neuron in the output layer fires and will be fed as observed label to the HMM. In this case, we have a deterministic decision, and the probability $p(y_n | \underline{x})$ describes what neuron $y_{n*}$ fires if vector $\underline{x}$ is presented to the input layer. Therefore, this probability reduces to

$$p(y_n | \underline{x}) = \begin{cases} 1 & \text{if} \quad y_n = y_{n*} \\ 0 & \text{else} \end{cases} \tag{7}$$

Then, (6) yields:

$$p(\underline{x} | \Omega) = \frac{p(\underline{x})}{p(\Omega)} \cdot p(\Omega | y_{n*}) \tag{8}$$

Now, the class-dependent probability $p(\underline{x} | \Omega)$ is expressed through the probability $p(\Omega | y_{n*})$, involving directly the firing neuron $y_{n*}$, when feature vector $\underline{x}$ is presented. One has now to turn back to (1), recalling the fact, that this equation describes the fact that the Markov models are trained with the ML criterion. It should also be recalled, that the entire sequence of feature vectors, $\underline{x}(k)$, $k=1...K$, results in a label stream of firing neurons $y_{n*}(k)$, $k=1...K$, where $y_{n*}(k)$ is the firing neuron if the k-th vector $\underline{x}(k)$ is presented to the neural network. Now, (8) can be substituted into (1) for each presentation k, yielding the modified ML criterion:

$$\underline{\theta}^* = \arg\max_{\underline{\theta}} \left\{ \sum_{k=1}^{K} \log \frac{p(\underline{x}(k))}{p(\Omega(k))} \cdot p(\Omega(k) | y_{n*}(k)) \right\}$$

$$= \arg\max_{\underline{\theta}} \left\{ \sum_{k=1}^{K} \log p(\underline{x}(k)) - \sum_{k=1}^{K} \log p(\Omega(k)) + \sum_{k=1}^{K} \log p(\Omega(k) | y_{n*}(k)) \right\} \tag{9}$$

Usually, in a continuous parameter system, the probability $p(\underline{x})$ can be expressed as:

$$p(\underline{x}) = \sum_{n=1}^{N} p(\underline{x} | y_n) \cdot p(y_n) \tag{10}$$

and is therefore dependent of the parameter vector $\underline{\theta}$, because in this case, $p(\underline{x} | y_n)$ can be interpreted as the probability provided by the Gaussian distributions, and the parameters of

the Gaussians will depend on $\underline{\theta}$. As just mentioned before, in a discrete system, only one firing neuron $y_{n*}$ survives, resulting in the fact that only the n*-th member remains in the sum in (10). This would correspond to only one "firing Gaussian" in the continuous case, leading to the following expression for $p(\underline{x})$:

$$p(\underline{x}) = p(\underline{x} \mid y_{n*}) \cdot p(y_{n*}) = p(\underline{x}, y_{n*}) = p(y_{n*} \mid \underline{x}) \cdot p(\underline{x}) \qquad (11)$$

Considering now the fact, that the acoustic processor is not represented by a Gaussian but instead by a vector quantizer, where the probability $p(y_{n*} \mid \underline{x})$ of the firing neuron is equal to 1, then (11) reduces to $p(\underline{x}) = p(\underline{x})$ and it becomes obvious that this probability is not affected by any distribution that depends on the parameter vector $\underline{\theta}$. This would be different, if $p(y_{n*} \mid \underline{x})$ in (11) would not have binary characteristics as in (7), but would be computed by a continuous function which in this case would depend on the parameter vector $\underline{\theta}$. Thus, without consideration of $p(\underline{x})$, the remaining expression to be maximized in (9) reduces to:

$$\underline{\theta}^* = \arg\max_{\underline{\theta}} \left[ -\sum_{k=1}^{K} \log p(\Omega(k)) + \sum_{k=1}^{K} \log p(\Omega(k) \mid y_{n*}(k)) \right] \qquad (12)$$

$$= \arg\max_{\underline{\theta}} \left[ -E\{\log p(\Omega)\} + E\{\log p(\Omega \mid y_{n*})\} \right]$$

These expectations of logarithmic probabilities are also defined as entropies. Therefore, (9) can be also written as

$$\underline{\theta}^* = \arg\max_{\underline{\theta}} \{H(\Omega) - H(\Omega \mid Y)\} \qquad (13)$$

This equation can be interpreted as follows: The term on the right side of (13) is also known as the mutual information $I(\Omega, Y)$ between the probabilistic variables $\Omega$ and Y, i.e.:

$$I(\Omega, Y) = H(\Omega) - H(\Omega \mid Y) = H(Y) - H(Y \mid \Omega) \qquad (14)$$

Therefore, the final information theory-based training criterion for the neural network can be formulated as follows: The synaptic weights of the neural network should be chosen as to maximize the mutual information between the string representing the classes of the vectors presented to the network input layer during training and the string representing the resulting sequence of firing neurons in the output layer of the neural network. This can be also expressed as the Maximum Mutual Information (MMI) criterion for neural network training. This concludes the proof that MMI neural networks are indeed optimal acoustic processors for HMM's trained with maximum likelihood principles.

## 3  REALIZATION OF MMI TRAINING ALGORITHMS FOR NEURAL NETWORKS

Training the synaptic weights of a neural network in order to achieve mutual information maximization is not easy. Two different algorithms have been developed for this task and can only be briefly outlined in this paper. A detailed description can be found in (Rigoll, 1994) and (Neukirchen, 1996). The first experiments used a single-layer neural network with Euclidean distance as propagation function. The first implementation of the MMI training paradigm has been realized in (Rigoll, 1994) and is based on a self-organizing procedure, starting with initial weights derived from k-means clustering of the training vectors, followed by an iterative procedure to modify the weights. The mutual information increases in a self-organizing way from a low value at the start to a much higher value after several iteration cycles. The second implementation has been realized

recently and is described in detail in (Neukirchen, 1996). It is based on the idea of using gradient methods for finding the MMI value. This technique has not been used before, because the maximum search for finding the firing neuron in the output layer has prevented the calculation of derivatives. This maximum search can be approximated using the softmax function, denoted as $s_n$ for the n-th neuron. It can be computed from the activations $z_l$ of all neurons as:

$$s_n = e^{z_n/T} / \sum_{l=1}^{N} e^{z_l/T}$$

(15)

where a small value for parameter T approximates a crisp maximum selection. Since the string $\Omega$ in (14) is always fixed during training and independent of the parameters in $\underline{\theta}$, only the function $H(\Omega|Y)$ has to be minimized. This function can also be expressed as

$$H(\Omega|Y) = - \sum_{m=1}^{M} \sum_{n=1}^{N} p(y_n, \Omega_m) \cdot \log p(\Omega_m | y_n)$$

$$= - \sum_{m=1}^{M} \sum_{n=1}^{N} p(y_n, \Omega_m) \cdot \log \frac{p(y_n, \Omega_m)}{\sum_{l=1}^{M} p(y_n, \Omega_l)} = - \sum_{m=1}^{M} \sum_{n=1}^{N} p^*[p(y_n, \Omega_m)]$$

(16)

A derivative with respect to a weight $w_{lj}$ of the neural network yields:

$$\frac{\partial H(\Omega|Y)}{\partial w_{lj}} = - \sum_{m=1}^{M} \sum_{n=1}^{N} \left\{ \frac{\partial p^*[p(y_n, \Omega_m)]}{\partial p(y_n, \Omega_m)} \cdot \frac{\partial p(y_n, \Omega_m)}{\partial s_n} \cdot \frac{\partial s_n}{\partial z_n} \cdot \frac{\partial z_n}{\partial w_{lj}} \right\}$$

(17)

As shown in (Neukirchen, 1996), all the required terms in (17) can be computed effectively and it is possible to realize a gradient descend method in order to maximize the mutual information of the training data. The great advantage of this method is the fact that it is now possible to generalize this algorithm for use in all popular neural network architectures, including multilayer and recurrent neural networks.

## 4 RESULTS FOR THE HYBRID SYSTEM

The new hybrid system has been developed and extensively tested using the Resource Management 1000 word speaker-independent continuous speech recognition task. First, a baseline discrete HMM system has been built up with all well-known features of a context-dependent HMM system. The performance of that baseline system is shown in column 2 of Table 1. The 1st column shows the performance of the hybrid system with the neural vector quantizer. This network has some special features not mentioned in the previous sections, e.g. it uses multiple frame input and has been trained on context-dependent classes. That means that the mutual information between the stream of firing neurons and the corresponding input stream of triphones has been maximized. In this way, the firing behavior of the network becomes sensitive to context-dependent units. Therefore, this network may be the only existing context-dependent acoustic processor, carrying the principle of triphone modeling from the HMM structure to the acoustic front end. It can be seen, that a substantially higher recognition performance is obtained with the hybrid system, that compares well with the leading continuous system (HTK, in column 3). It is expected, that the system will be further improved in the near future through various additional features, including full exploitation of multilayer neural VQ's

and several conventional HMM improvements, e.g. the use of crossword triphones. Recent results on the larger Wall Street Journal (WSJ) database have shown a 10.5% error rate for the hybrid system compared to a 13.4% error rate for a standard discrete system, using the 5k vocabulary test with bigram language model of perplexity 110. This error rate can be further reduced to 8.9% using crossword triphones and 6.6% with a trigram language model. This rate compares already quite favorably with the best continuous systems for the same task. It should be noted that this hybrid WSJ system is still in its initial stage and the neural component is not yet as sophisticated as in the RM system.

## 5 CONCLUSION

A new neural network paradigm and the resulting hybrid HMM/ANN speech recognition system have been presented in this paper. The new approach performs already very well and is still perfectible. It gains its good performance from the following facts: (1) The use of information theory-based training algorithms for the neural vector quantizer, which can be shown to be optimal for the hybrid approach. (2) The possibility of introducing context-dependency not only to the HMM's, but also to the neural quantizer. (3) The fact that this hybrid approach allows the combination of an optimal neural acoustic processor with the most advanced context-dependent HMM system. We will continue to further implement various possible improvements for our hybrid speech recognition system.

## REFERENCES

Rigoll, G. (1994) Maximum Mutual Information Neural Networks for Hybrid Connectionist-HMM Speech Recognition Systems, *IEEE Transactions on Speech and Audio Processing*, Vol. 2, No. 1, Special Issue on Neural Networks for Speech Processing, pp. 175-184

Neukirchen, C. & Rigoll, G. (1996) Training of MMI Neural Networks as Vector Quantizers, *Internal Report, Gerhard-Mercator-University Duisburg, Faculty of Electrical Engineering*, available via http://www.fb9-ti.uni-duisburg.de/veroeffentl.html

Bourlard, H. & Morgan, N. (1994) *Connectionist Speech Recognition: A Hybrid Approach*, Kluwer Academic Publishers

Hochberg, M., Renals, S., Robinson, A., Cook, G. (1995) Recent Improvements to the ABBOT Large Vocabulary CSR System, *in Proc. IEEE-ICASSP*, Detroit, pp. 69-72

Rigoll, G., Neukirchen, C., Rottland, J. (1996) A New Hybrid System Based on MMI-Neural Networks for the RM Speech Recognition Task, *in Proc. IEEE-ICASSP*, Atlanta

Table 1: Comparison of recognition rates for different speech recognition systems

| RM SI word recognition rate with word pair grammar: correctness (accuracy) | | | |
|---|---|---|---|
| test set | hybrid MMI-NN system | baseline k-means VQ system | continuous pdf system (HTK) |
| Feb.'89 | **96,3 % (95,6 %)** | 94,3 % (93,6 %) | 96,0 % (95,5 %) |
| Oct.'89 | **95,4 % (94,5 %)** | 93,5 % (92,0 %) | 95,4 % (94,9 %) |
| Feb.'91 | **96,7 % (95,9 %)** | 94,4 % (93,5 %) | 96,6 % (96,0 %) |
| Sep.'92 | **93,9 % (92,5 %)** | 90,7 % (88,9 %) | 93,6 % (92,6 %) |
| average | **95,6 % (94,6 %)** | 93,2 % (92,0 %) | 95,4 % (94,7 %) |